# Fixing Max-Product: Convergent Message Passing Algorithms for MAP LP-Relaxations

**Amir Globerson    Tommi Jaakkola**
Computer Science and Artificial Intelligence Laboratory
Massachusetts Institute of Technology
Cambridge, MA 02139
gamir,tommi@csail.mit.edu

## Abstract

We present a novel message passing algorithm for approximating the MAP problem in graphical models. The algorithm is similar in structure to max-product but unlike max-product it always converges, and can be proven to find the exact MAP solution in various settings. The algorithm is derived via block coordinate descent in a dual of the LP relaxation of MAP, but does not require any tunable parameters such as step size or tree weights. We also describe a generalization of the method to cluster based potentials. The new method is tested on synthetic and real-world problems, and compares favorably with previous approaches.

Graphical models are an effective approach for modeling complex objects via local interactions. In such models, a distribution over a set of variables is assumed to factor according to cliques of a graph with potentials assigned to each clique. Finding the assignment with highest probability in these models is key to using them in practice, and is often referred to as the MAP (maximum aposteriori) assignment problem. In the general case the problem is NP hard, with complexity exponential in the tree-width of the underlying graph.

Linear programming (LP) relaxations have proven very useful in approximating the MAP problem, and often yield satisfactory empirical results. These approaches relax the constraint that the solution is integral, and generally yield non-integral solutions. However, when the LP solution is integral, it is guaranteed to be the exact MAP. For some classes of problems the LP relaxation is provably correct. These include the minimum cut problem and maximum weight matching in bi-partite graphs [8]. Although LP relaxations can be solved using standard LP solvers, this may be computationally intensive for large problems [13]. The key problem with generic LP solvers is that they do not use the graph structure explicitly and thus may be sub-optimal in terms of computational efficiency.

The max-product method [7] is a message passing algorithm that is often used to approximate the MAP problem. In contrast to generic LP solvers, it makes direct use of the graph structure in constructing and passing messages, and is also very simple to implement. The relation between max-product and the LP relaxation has remained largely elusive, although there are some notable exceptions: For tree-structured graphs, max-product and LP both yield the exact MAP. A recent result [1] showed that for maximum weight matching on bi-partite graphs max-product and LP also yield the exact MAP [1]. Finally, Tree-Reweighted max-product (TRMP) algorithms [5, 10] were shown to converge to the LP solution for binary $x_i$ variables, as shown in [6].

In this work, we propose the Max Product Linear Programming algorithm (MPLP) - a very simple variation on max-product that is guaranteed to converge, and has several advantageous properties. MPLP is derived from the dual of the LP relaxation, and is equivalent to block coordinate descent in the dual. Although this results in monotone improvement of the dual objective, global convergence is not always guaranteed since coordinate descent may get stuck in suboptimal points. This can be remedied using various approaches, but in practice we have found MPLP to converge to the LP

solution in a majority of the cases we studied. To derive MPLP we use a special form of the dual LP, which involves the introduction of redundant primal variables and constraints. We show how the dual variables corresponding to these constraints turn out to be the *messages* in the algorithm. We evaluate the method on Potts models and protein design problems, and show that it compares favorably with max-product (which often does not converge for these problems) and TRMP.

## 1   The Max-Product and MPLP Algorithms

The max-product algorithm [7] is one of the most often used methods for solving MAP problems. Although it is neither guaranteed to converge to the correct solution, or in fact converge at all, it provides satisfactory results in some cases. Here we present two algorithms: EMPLP (edge based MPLP) and NMPLP (node based MPLP), which are structurally very similar to max-product, but have several key advantages:

- After each iteration, the messages yield an upper bound on the MAP value, and the sequence of bounds is monotone decreasing and convergent. The messages also have a limit point that is a fixed point of the update rule.

- No additional parameters (e.g., tree weights as in [6]) are required.

- If the fixed point beliefs have a unique maximizer then they correspond to the exact MAP.

- For binary variables, MPLP can be used to obtain the solution to an LP relaxation of the MAP problem. Thus, when this LP relaxation is exact and variables are binary, MPLP will find the MAP solution. Moreover, for any variable whose beliefs are not tied, the MAP assignment can be found (i.e., the solution is partially decodable).

Pseudo code for the algorithms (and for max-product) is given in Fig. 1. As we show in the next sections, MPLP is essentially a block coordinate descent algorithm in the dual of a MAP LP relaxation. Every update of the MPLP messages corresponds to exact minimization of a set of dual variables. For EMPLP minimization is over the set of variables corresponding to an edge, and for NMPLP it is over the set of variables corresponding to all the edges a given node appears in (i.e., a star). The properties of MPLP result from its relation to the LP dual. In what follows we describe the derivation of the MPLP algorithms and prove their properties.

## 2   The MAP Problem and its LP Relaxation

We consider functions over $n$ variables $\boldsymbol{x} = \{x_1, \ldots, x_n\}$ defined as follows. Given a graph $G = (V, E)$ with $n$ vertices, and potentials $\theta_{ij}(x_i, x_j)$ for all edges $ij \in E$, define the function[1]

$$f(\boldsymbol{x}; \boldsymbol{\theta}) = \sum_{ij \in E} \theta_{ij}(x_i, x_j) \,. \tag{1}$$

The MAP problem is defined as finding an assignment $\boldsymbol{x}_M$ that maximizes the function $f(\boldsymbol{x}; \boldsymbol{\theta})$. Below we describe the standard LP relaxation for this problem. Denote by $\{\mu_{ij}(x_i, x_j)\}_{ij \in E}$ distributions over variables corresponding to edges $ij \in E$ and $\{\mu_i(x_i)\}_{i \in V}$ distributions corresponding to nodes $i \in V$. We will use $\boldsymbol{\mu}$ to denote a given set of distributions over all edges and nodes. The set $\mathcal{M}_L(G)$ is defined as the set of $\boldsymbol{\mu}$ where pairwise and singleton distributions are consistent

$$\mathcal{M}_L(G) = \left\{ \boldsymbol{\mu} \geq 0 \, \middle| \, \begin{array}{ll} \sum_{\hat{x}_i} \mu_{ij}(\hat{x}_i, x_j) = \mu_j(x_j) \,, & \sum_{\hat{x}_j} \mu_{ij}(x_i, \hat{x}_j) = \mu_i(x_i) \quad \forall ij \in E, x_i, x_j \\ \sum_{x_i} \mu_i(x_i) = 1 & \forall i \in V \end{array} \right\}$$

Now consider the following linear program:

$$\underline{\text{MAPLPR}}: \qquad \boldsymbol{\mu}^{L*} = \arg \max_{\boldsymbol{\mu} \in \mathcal{M}_L(G)} \boldsymbol{\mu} \cdot \boldsymbol{\theta} \,. \tag{2}$$

where $\boldsymbol{\mu} \cdot \boldsymbol{\theta}$ is shorthand for $\boldsymbol{\mu} \cdot \boldsymbol{\theta} = \sum_{ij \in E} \sum_{x_i, x_j} \theta_{ij}(x_i, x_j) \mu_{ij}(x_i, x_j)$. It is easy to show (see e.g., [10]) that the optimum of MAPLPR yields an upper bound on the MAP value, i.e. $\boldsymbol{\mu}^{L*} \cdot \boldsymbol{\theta} \geq f(\boldsymbol{x}_M)$. Furthermore, when the optimal $\mu_i(x_i)$ have only integral values, the assignment that maximizes $\mu_i(x_i)$ yields the correct MAP assignment. In what follows we show how the MPLP algorithms can be derived from the dual of MAPLPR.

## 3 The LP Relaxation Dual

Since MAPLPR is an LP, it has an equivalent convex dual. In App. A we derive a special dual of MAPLPR using a different representation of $\mathcal{M}_L(G)$ with redundant variables. The advantage of this dual is that it allows the derivation of simple message passing algorithms. The dual is described in the following proposition.

**Proposition 1** *The following optimization problem is a convex dual of MAPLPR*

$$\underline{DMAPLPR}:$$
$$\min \qquad \sum_i \max_{x_i} \sum_{k \in N(i)} \max_{x_k} \beta_{ki}(x_k, x_i) \tag{3}$$
$$s.t. \qquad \beta_{ji}(x_j, x_i) + \beta_{ij}(x_i, x_j) = \theta_{ij}(x_i, x_j),$$

*where the dual variables are $\beta_{ij}(x_i, x_j)$ for all $ij, ji \in E$ and values of $x_i$ and $x_j$.*

The dual has an intuitive interpretation in terms of re-parameterizations. Consider the *star* shaped graph $G_i$ consisting of node $i$ and all its neighbors $N(i)$. Assume the potential on edge $ki$ (for $k \in N(i)$) is $\beta_{ki}(x_k, x_i)$. The value of the MAP assignment for this model is $\max_{x_i} \sum_{k \in N(i)} \max_{x_k} \beta_{ki}(x_k, x_i)$. This is exactly the term in the objective of DMAPLPR. Thus the dual corresponds to individually decoding star graphs around all nodes $i \in V$ where the potentials on the graph edges should sum to the original potential. It is easy to see that this will always result in an upper bound on the MAP value. The somewhat surprising result of the duality is that there exists a $\boldsymbol{\beta}$ assignment such that *star decoding* yields the optimal value of MAPLPR.

## 4 Block Coordinate Descent in the Dual

To obtain a convergent algorithm we use a simple block coordinate descent strategy. At every iteration, fix all variables except a subset, and optimize over this subset. It turns out that this can be done in closed form for the cases we consider. We begin by deriving the EMPLP algorithm. Consider fixing all the $\boldsymbol{\beta}$ variables except those corresponding to some edge $ij \in E$ (i.e., $\beta_{ij}$ and $\beta_{ji}$), and minimizing DMAPLPR over the non-fixed variables. Only two terms in the DMAPLPR objective depend on $\beta_{ij}$ and $\beta_{ji}$. We can write those as

$$f(\beta_{ij}, \beta_{ji}) = \max_{x_i} \left[ \lambda_i^{-j}(x_i) + \max_{x_j} \beta_{ji}(x_j, x_i) \right] + \max_{x_i} \left[ \lambda_j^{-i}(x_j) + \max_{x_i} \beta_{ij}(x_i, x_j) \right] \tag{4}$$

where we defined $\lambda_i^{-j}(x_i) = \sum_{k \in N(i) \backslash j} \lambda_{ki}(x_i)$ and $\lambda_{ki}(x_i) = \max_{x_k} \beta_{ki}(x_k, x_i)$ as in App. A. Note that the function $f(\beta_{ij}, \beta_{ji})$ depends on the other $\boldsymbol{\beta}$ values only through $\lambda_j^{-i}(x_j)$ and $\lambda_i^{-j}(x_i)$. This implies that the optimization can be done solely in terms of $\lambda_{ij}(x_j)$ and there is no need to store the $\beta$ values explicitly. The optimal $\beta_{ij}, \beta_{ji}$ are obtained by minimizing $f(\beta_{ij}, \beta_{ji})$ subject to the *re-parameterization* constraint $\beta_{ji}(x_j, x_i) + \beta_{ij}(x_i, x_j) = \theta_{ij}(x_i, x_j)$. The following proposition characterizes the minimum of $f(\beta_{ij}, \beta_{ji})$. In fact, as mentioned above, we do not need to characterize the optimal $\beta_{ij}(x_i, x_j)$ itself, but only the new $\lambda$ values.

**Proposition 2** *Maximizing the function $f(\beta_{ij}, \beta_{ji})$ yields the following $\lambda_{ji}(x_i)$ (and the equivalent expression for $\lambda_{ij}(x_j)$)*

$$\lambda_{ji}(x_i) = -\frac{1}{2}\lambda_i^{-j}(x_i) + \frac{1}{2}\max_{x_j}\left[\lambda_j^{-i}(x_j) + \theta_{ij}(x_i, x_j)\right]$$

The proposition is proved in App. B. The $\lambda$ updates above result in the EMPLP algorithm, described in Fig. 1. Note that since the $\boldsymbol{\beta}$ optimization affects both $\lambda_{ji}(x_i)$ and $\lambda_{ij}(x_j)$, both these *messages* need to be updated simultaneously.

We proceed to derive the NMPLP algorithm. For a given node $i \in V$, we consider all its neighbors $j \in N(i)$, and wish to optimize over the variables $\beta_{ji}(x_j, x_i)$ for $ji, ij \in E$ (i.e., all the edges in a *star* centered on $i$), while the other variables are fixed. One way of doing so is to use the EMPLP algorithm for the edges in the star, and iterate it until convergence. We now show that the result of

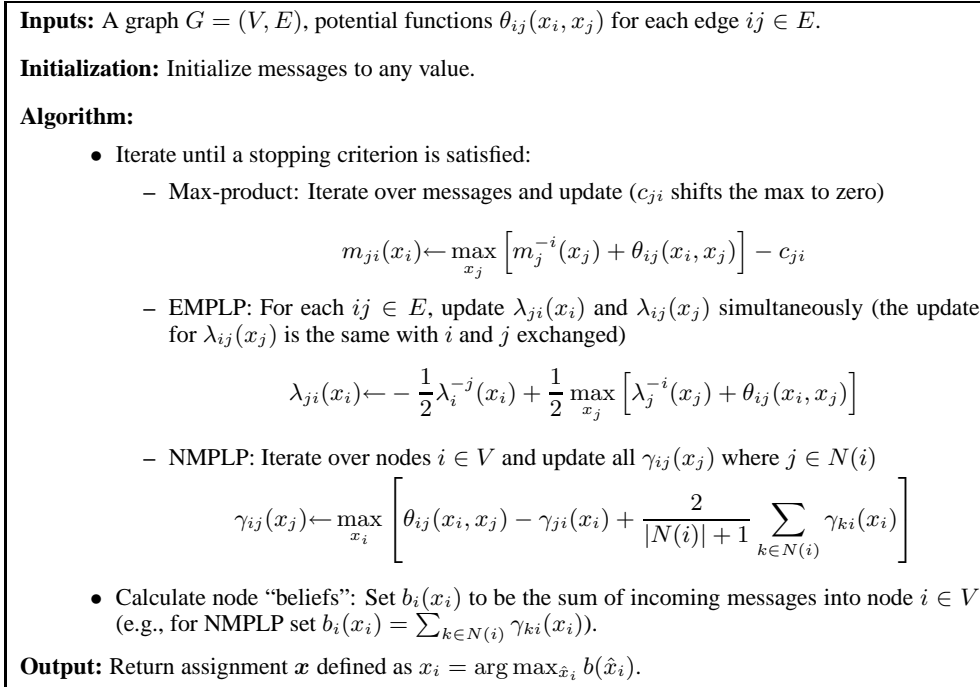

**Inputs:** A graph $G = (V, E)$, potential functions $\theta_{ij}(x_i, x_j)$ for each edge $ij \in E$.

**Initialization:** Initialize messages to any value.

**Algorithm:**

- Iterate until a stopping criterion is satisfied:
    - Max-product: Iterate over messages and update ($c_{ji}$ shifts the max to zero)

    $$m_{ji}(x_i) \leftarrow \max_{x_j} \left[ m_j^{-i}(x_j) + \theta_{ij}(x_i, x_j) \right] - c_{ji}$$

    - EMPLP: For each $ij \in E$, update $\lambda_{ji}(x_i)$ and $\lambda_{ij}(x_j)$ simultaneously (the update for $\lambda_{ij}(x_j)$ is the same with $i$ and $j$ exchanged)

    $$\lambda_{ji}(x_i) \leftarrow -\frac{1}{2}\lambda_i^{-j}(x_i) + \frac{1}{2}\max_{x_j}\left[ \lambda_j^{-i}(x_j) + \theta_{ij}(x_i, x_j) \right]$$

    - NMPLP: Iterate over nodes $i \in V$ and update all $\gamma_{ij}(x_j)$ where $j \in N(i)$

    $$\gamma_{ij}(x_j) \leftarrow \max_{x_i} \left[ \theta_{ij}(x_i, x_j) - \gamma_{ji}(x_i) + \frac{2}{|N(i)|+1} \sum_{k \in N(i)} \gamma_{ki}(x_i) \right]$$

- Calculate node "beliefs": Set $b_i(x_i)$ to be the sum of incoming messages into node $i \in V$ (e.g., for NMPLP set $b_i(x_i) = \sum_{k \in N(i)} \gamma_{ki}(x_i)$).

**Output:** Return assignment $\boldsymbol{x}$ defined as $x_i = \arg\max_{\hat{x}_i} b(\hat{x}_i)$.

Figure 1: The max-product, EMPLP and NMPLP algorithms. Max-product, EMPLP and NMPLP use messages $m_{ij}$, $\lambda_{ij}$ and $\gamma_{ij}$ respectively. We use the notation $m_j^{-i}(x_j) = \sum_{k \in N(j) \setminus i} m_{kj}(x_j)$.

this optimization can be found in closed form. The assumption about $\boldsymbol{\beta}$ being fixed outside the star implies that $\lambda_j^{-i}(x_j)$ is fixed. Define: $\gamma_{ji}(x_i) = \max_{x_j}\left[\theta_{ij}(x_i, x_j) + \lambda_j^{-i}(x_j)\right]$. Simple algebra yields the following relation between $\lambda_i^{-j}(x_i)$ and $\gamma_{ki}(x_i)$ for $k \in N(i)$

$$\lambda_i^{-j}(x_i) = -\gamma_{ji}(x_i) + \frac{2}{|N(i)|+1} \sum_{k \in N(i)} \gamma_{ki}(x_i) \tag{5}$$

Plugging this into the definition of $\gamma_{ji}(x_i)$ we obtain the NMPLP update in Fig. 1. The messages for both algorithms can be initialized to any value since it can be shown that after one iteration they will correspond to valid $\boldsymbol{\beta}$ values.

## 5  Convergence Properties

The MPLP algorithm decreases the dual objective (i.e., an upper bound on the MAP value) at every iteration, and thus its dual objective values form a convergent sequence. Using arguments similar to [5] it can be shown that MPLP has a limit point that is a fixed point of its updates. This in itself does not guarantee convergence to the dual optimum since coordinate descent algorithms may get stuck at a point that is not a global optimum. There are ways of overcoming this difficulty, for example by smoothing the objective [4] or using techniques as in [2] (see p. 636). We leave such extensions for further work. In this section we provide several results about the properties of the MPLP fixed points and their relation to the corresponding LP. First, we claim that if all beliefs have unique maxima then the *exact* MAP assignment is obtained.

**Proposition 3** *If the fixed point of MPLP has $b_i(x_i)$ such that for all $i$ the function $b_i(x_i)$ has a unique maximizer $x_i^*$, then $\boldsymbol{x}^*$ is the solution to the MAP problem and the LP relaxation is exact.*

Since the dual objective is always greater than or equal to the MAP value, it suffices to show that there exists a dual feasible point whose objective value is $f(x^*)$. Denote by $\beta^*, \lambda^*$ the value of the corresponding dual parameters at the fixed point of MPLP. Then the dual objective satisfies

$$\sum_i \max_{x_i} \sum_{k \in N(i)} \lambda_{ki}^*(x_i) = \sum_i \sum_{k \in N(i)} \max_{x_k} \beta_{ki}^*(x_k, x_i^*) = \sum_i \sum_{k \in N(i)} \beta_{ki}^*(x_k^*, x_i^*) = f(\boldsymbol{x}^*)$$

To see why the second equality holds, note that $b_i(x_i^*) = \max_{x_i,x_j} \lambda_i^{-j}(x_i) + \beta_{ji}(x_j, x_i)$ and $b_j(x_j^*) = \max_{x_i,x_j} \lambda_j^{-i}(x_j) + \beta_{ij}(x_i, x_j)$. By the equalization property in Eq. 9 the arguments of the two max operations are equal. From the unique maximum assumption it follows that $x_i^*, x_j^*$ are the unique maximizers of the above. It follows that $\beta_{ji}, \beta_{ij}$ are also maximized by $x_i^*, x_j^*$.

In the general case, the MPLP fixed point may not correspond to a primal optimum because of the local optima problem with coordinate descent. However, when the variables are binary, fixed points do correspond to primal solutions, as the following proposition states.

**Proposition 4** *When $x_i$ are binary, the MPLP fixed point can be used to obtain the primal optimum.*

The claim can be shown by constructing a primal optimal solution $\boldsymbol{\mu}^*$. For tied $b_i$, set $\mu_i^*(x_i)$ to 0.5 and for untied $b_i$, set $\mu_i^*(x_i^*)$ to 1. If $b_i, b_j$ are not tied we set $\mu_{ij}^*(x_i^*, x_j^*) = 1$. If $b_i$ is not tied but $b_j$ is, we set $\mu_{ij}^*(x_i^*, x_j) = 0.5$. If $b_i, b_j$ are tied then $\beta_{ji}, \beta_{ij}$ can be shown to be maximized at either $x_i^*, x_j^* = (0,0), (1,1)$ or $x_i^*, x_j^* = (0,1), (1,0)$. We then set $\mu_{ij}^*$ to be 0.5 at one of these assignment pairs. The resulting $\boldsymbol{\mu}^*$ is clearly primal feasible. Setting $\delta_i^* = b_i^*$ we obtain that the dual variables $(\delta^*, \lambda^*, \beta^*)$ and primal $\boldsymbol{\mu}^*$ satisfy complementary slackness for the LP in Eq. 7 and therefore $\boldsymbol{\mu}^*$ is primal optimal. The binary optimality result implies partial decodability, since [6] shows that the LP is partially decodable for binary variables.

## 6  Beyond pairwise potentials: Generalized MPLP

In the previous sections we considered maximizing functions which factor according to the edges of the graph. A more general setting considers clusters $c_1, \ldots, c_k \subset \{1, \ldots, n\}$ (the set of clusters is denoted by $\mathcal{C}$), and a function $f(\boldsymbol{x}; \boldsymbol{\theta}) = \sum_c \theta_c(x_c)$ defined via potentials over clusters $\theta_c(x_c)$. The MAP problem in this case also has an LP relaxation (see e.g. [11]). To define the LP we introduce the following definitions: $\mathcal{S} = \{c \cap \hat{c} : c, \hat{c} \in \mathcal{C}, c \cap \hat{c} \neq \emptyset\}$ is the set of intersection between clusters and $\mathcal{S}(c) = \{s \in \mathcal{S} : s \subseteq c\}$ is the set of overlap sets for cluster $c$. We now consider marginals over the variables in $c \in \mathcal{C}$ and $s \in \mathcal{S}$ and require that cluster marginals *agree* on their overlap. Denote this set by $\mathcal{M}_L(\mathcal{C})$. The LP relaxation is then to maximize $\boldsymbol{\mu} \cdot \boldsymbol{\theta}$ subject to $\boldsymbol{\mu} \in \mathcal{M}_L(\mathcal{C})$.

As in Sec. 4, we can derive message passing updates that result in monotone decrease of the dual LP of the above relaxation. The derivation is similar and we omit the details. The key observation is that one needs to introduce $|\mathcal{S}(c)|$ copies of each marginal $\mu_c(x_c)$ (instead of the two copies in the pairwise case). Next, as in the EMPLP derivation we assume all $\boldsymbol{\beta}$ are fixed except those corresponding to some cluster $c$. The resulting messages are $\lambda_{c \to s}(x_s)$ from a cluster $c$ to all of its intersection sets $s \in \mathcal{S}(c)$. The update on these messages turns out to be:

$$\lambda_{c \to s}(x_s) = -\left(1 - \frac{1}{|\mathcal{S}(c)|}\right) \lambda_s^{-c}(x_s) + \frac{1}{|\mathcal{S}(c)|} \max_{x_{c \setminus s}} \left[ \sum_{\hat{s} \in \mathcal{S}(c) \setminus s} \lambda_{\hat{s}}^{-c}(x_{\hat{s}}) + \theta_c(x_c) \right]$$

where for a given $c \in \mathcal{C}$ all $\lambda_{c \to s}$ should be updated simultaneously for $s \in \mathcal{S}(c)$, and $\lambda_s^{-c}(x_s)$ is defined as the sum of messages into $s$ that are not from $c$. We refer to this algorithm as Generalized EMPLP (GEMPLP). It is possible to derive an algorithm similar to NMPLP that updates several clusters simultaneously, but its structure is more involved and we do not address it here.

## 7  Related Work

Weiss et al. [11] recently studied the fixed points of a class of *max-product like* algorithms. Their analysis focused on properties of fixed points rather than convergence guarantees. Specifically, they showed that if the counting numbers used in a generalized max-product algorithm satisfy certain properties, then its fixed points will be the exact MAP if the beliefs have unique maxima, and for binary variables the solution can be partially decodable. Both these properties are obtained for the MPLP fixed points, and in fact we can show that MPLP satisfies the conditions in [11], so that we obtain these properties as corollaries of [11]. We stress however, that [11] does not address convergence of algorithms, but rather properties of their fixed points, if they converge.

MPLP is similar in some aspects to Kolmogorov's TRW-S algorithm [5]. TRW-S is also a monotone coordinate descent method in a dual of the LP relaxation and its fixed points also have similar

guarantees to those of MPLP [6]. Furthermore, convergence to a local optimum may occur, as it does for MPLP. One advantage of MPLP lies in the simplicity of its updates and the fact that it is parameter free. The other is its simple generalization to potentials over clusters of nodes (Sec. 6). Recently, several new dual LP algorithms have been introduced, which are more closely related to our formalism. Werner [12] presented a class of algorithms which also improve the dual LP at every iteration. The simplest of those is the max-sum-diffusion algorithm, which is similar to our EMPLP algorithm, although the updates are different from ours. Independently, Johnson *et al.* [4] presented a class of algorithms that improve duals of the MAP-LP using coordinate descent. They decompose the model into tractable parts by replicating variables and enforce replication constraints within the Lagrangian dual. Our basic formulation in Eq. 3 could be derived from their perspective. However, the updates in the algorithm and the analysis differ. Johnson *et al.* also presented a method for overcoming the local optimum problem, by smoothing the objective so that it is strictly convex. Such an approach could also be used within our algorithms. Vontobel and Koetter [9] recently introduced a coordinate descent algorithm for decoding LDPC codes. Their method is specifically tailored for this case, and uses updates that are similar to our edge based updates.

Finally, the concept of dual coordinate descent may be used in approximating marginals as well. In [3] we use such an approach to optimize a variational bound on the partition function. The derivation uses some of the ideas used in the MPLP dual, but importantly does not find the minimum for each coordinate. Instead, a *gradient like* step is taken at every iteration to decrease the dual objective.

## 8 Experiments

We compared NMPLP to three other message passing algorithms:[2] Tree-Reweighted max-product (TRMP) [10],[3] standard max-product (MP), and GEMPLP. For MP and TRMP we used the standard approach of damping messages using a factor of $\alpha = 0.5$. We ran all algorithms for a maximum of 2000 iterations, and used the *hit-time* measure to compare their speed of convergence. This measure is defined as follows: At every iteration the beliefs can be used to obtain an assignment $x$ with value $f(x)$. We define the *hit-time* as the first iteration at which the maximum value of $f(x)$ is achieved.[4]

We first experimented with a $10 \times 10$ grid graph, with 5 values per state. The function $f(x)$ was a Potts model: $f(x) = \sum_{ij \in E} \theta_{ij} \mathcal{I}(x_i = x_j) + \sum_{i \in V} \theta_i(x_i)$.[5] The values for $\theta_{ij}$ and $\theta_i(x_i)$ were randomly drawn from $[-c_I, c_I]$ and $[-c_F, c_F]$ respectively, and we used values of $c_I$ and $c_F$ in the range range $[0.1, 2.35]$ (with intervals of $0.25$), resulting in $100$ different models. The clusters for GEMPLP were the faces of the graph [14]. To see if NMPLP converges to the LP solution we also used an LP solver to solve the LP relaxation. We found that the the normalized difference between NMPLP and LP objective was at most $10^{-3}$ (median $10^{-7}$), suggesting that NMPLP typically converged to the LP solution. Fig. 2 (top row) shows the results for the three algorithms. It can be seen that while all non-cluster based algorithms obtain similar $f(x)$ values, NMPLP has better *hit-time* (in the median) than TRMP and MP, and MP does not converge in many cases (see caption). GEMPLP converges more slowly than NMPLP, but obtains much better $f(x)$ values. In fact, in $99\%$ of the cases the normalized difference between the GEMPLP objective and the $f(x)$ value was less than $10^{-5}$, suggesting that the exact MAP solution was found.

We next applied the algorithms to the real world problems of protein design. In [13], Yanover et al. show how these problems can be formalized in terms of finding a MAP in an appropriately constructed graphical model.[6] We used all algorithms except GNMPLP (since there is no natural choice for clusters in this case) to approximate the MAP solution on the 97 models used in [13]. In these models the number of states per variable is $2 - 158$, and there are up to $180$ variables per model. Fig. 2 (bottom) shows results for all the design problems. In this case only $11\%$ of the MP runs converged, and NMPLP was better than TRMP in terms of *hit-time* and comparable in $f(x)$ value. The performance of MP was good on the runs where it converged.

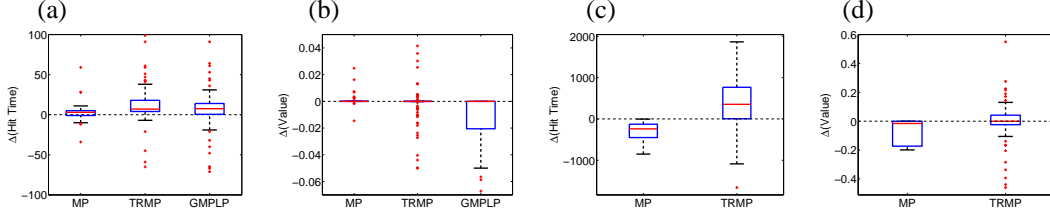

Figure 2: Evaluation of message passing algorithms on Potts models and protein design problems. **(a,c)**: Convergence time results for the Potts models (a) and protein design problems (c). The box-plots (horiz. red line indicates median) show the difference between the *hit-time* for the other algorithms and NMPLP. **(b,d)**: Value of integer solutions for the Potts models (b) and protein design problems (d). The box-plots show the normalized difference between the value of $f(\boldsymbol{x})$ for NMPLP and the other algorithms. All figures are such that better MPLP performance yields positive $Y$ axis values. Max-product converged on $58\%$ of the cases for the Potts models, and on $11\%$ of the protein problems. Only convergent max-product runs are shown.

## 9    Conclusion

We have presented a convergent algorithm for MAP approximation that is based on block coordinate descent of the MAP-LP relaxation dual. The algorithm can also be extended to cluster based functions, which result empirically in improved MAP estimates. This is in line with the observations in [14] that generalized belief propagation algorithms can result in significant performance improvements. However generalized max-product algorithms [14] are not guaranteed to converge whereas GMPLP is. Furthermore, the GMPLP algorithm does not require a region graph and only involves intersection between pairs of clusters. In conclusion, MPLP has the advantage of resolving the convergence problems of max-product while retaining its simplicity, and offering the theoretical guarantees of LP relaxations. We thus believe it should be useful in a wide array of applications.

## A    Derivation of the dual

Before deriving the dual, we first express the constraint set $\mathcal{M}_L(G)$ in a slightly different way. The definition of $\mathcal{M}_L(G)$ in Sec. 2 uses a single distribution $\mu_{ij}(x_i, x_j)$ for every $ij \in E$. In what follows, we use *two* copies of this pairwise distribution for every edge, which we denote $\bar{\mu}_{ij}(x_i, x_j)$ and $\bar{\mu}_{ji}(x_j, x_i)$, and we add the constraint that these two copies both equal the *original* $\mu_{ij}(x_i, x_j)$. For this extended set of pairwise marginals, we consider the following set of constraints which is clearly equivalent to $\mathcal{M}_L(G)$. On the rightmost column we give the dual variables that will correspond to each constraint (we omit non-negativity constraints).

$$
\begin{array}{llll}
\bar{\mu}_{ij}(x_i, x_j) = \mu_{ij}(x_i, x_j) & \forall ij \in E, x_i, x_j & \beta_{ij}(x_i, x_j) \\
\bar{\mu}_{ji}(x_j, x_i) = \mu_{ij}(x_i, x_j) & \forall ij \in E, x_i, x_j & \beta_{ji}(x_j, x_i) \\
\sum_{\hat{x}_i} \bar{\mu}_{ij}(\hat{x}_i, x_j) = \mu_j(x_j) & \forall ij \in E, x_j & \lambda_{ij}(x_j) \\
\sum_{\hat{x}_j} \bar{\mu}_{ji}(\hat{x}_j, x_i) = \mu_i(x_i) & \forall ji \in E, x_i & \lambda_{ji}(x_i) \\
\sum_{x_i} \mu_i(x_i) = 1 & \forall i \in V & \delta_i
\end{array}
\tag{6}
$$

We denote the set of $(\boldsymbol{\mu}, \bar{\boldsymbol{\mu}})$ satisfying these constraints by $\bar{\mathcal{M}}_L(G)$. We can now state an LP that is equivalent to MAPLPR, only with an extended set of variables and constraints. The equivalent problem is to maximize $\boldsymbol{\mu} \cdot \boldsymbol{\theta}$ subject to $(\boldsymbol{\mu}, \bar{\boldsymbol{\mu}}) \in \bar{\mathcal{M}}_L(G)$ (note that the objective uses the *original* $\boldsymbol{\mu}$ copy). LP duality transformation of the extended problem yields the following LP

$$
\begin{array}{lll}
\min & \sum_i \delta_i & \\
s.t. & \lambda_{ij}(x_j) - \beta_{ij}(x_i, x_j) \geq 0 & \forall ij, ji \in E, x_i, x_j \\
& \beta_{ij}(x_i, x_j) + \beta_{ji}(x_j, x_i) = \theta_{ij}(x_i, x_j) & \forall ij \in E, x_i, x_j \\
& -\sum_{k \in N(i)} \lambda_{ki}(x_i) + \delta_i \geq 0 & \forall i \in V, x_i
\end{array}
\tag{7}
$$

We next simplify the above LP by eliminating some of its constraints and variables. Since each variable $\delta_i$ appears in only one constraint, and the objective minimizes $\delta_i$ it follows that $\delta_i = \max_{x_i} \sum_{k \in N(i)} \lambda_{ki}(x_i)$ and the constraints with $\delta_i$ can be discarded. Similarly, since $\lambda_{ij}(x_j)$ appears in a single constraint, we have that for all $ij \in E, ji \in E, x_i, x_j$ $\lambda_{ij}(x_j) = \max_{x_i} \beta_{ij}(x_i, x_j)$ and the constraints with $\lambda_{ij}(x_j), \lambda_{ji}(x_i)$ can also be discarded. Using the eliminated $\delta_i$ and $\lambda_{ji}(x_i)$

variables, we obtain that the LP in Eq. 7 is equivalent to that in Eq. 3. Note that the objective in Eq. 3 is convex since it is a sum of point-wise maxima of convex functions.

## B   Proof of Proposition 2

We wish to minimize $f$ in Eq. 4 subject to the constraint that $\beta_{ij} + \beta_{ji} = \theta_{ij}$. Rewrite $f$ as

$$f(\beta_{ij}, \beta_{ji}) = \max_{x_i, x_j} \left[ \lambda_i^{-j}(x_i) + \beta_{ji}(x_j, x_i) \right] + \max_{x_i, x_j} \left[ \lambda_j^{-i}(x_j) + \beta_{ij}(x_i, x_j) \right] \tag{8}$$

The sum of the two arguments in the max is $\lambda_i^{-j}(x_i) + \lambda_j^{-i}(x_j) + \theta_{ij}(x_i, x_j)$ (because of the constraints on $\boldsymbol{\beta}$). Thus the minimum must be greater than $\frac{1}{2} \max_{x_i, x_j} \left[ \lambda_i^{-j}(x_i) + \lambda_j^{-i}(x_j) + \theta_{ij}(x_i, x_j) \right]$. One assignment to $\boldsymbol{\beta}$ that achieves this minimum is obtained by requiring an equalization condition:[7]

$$\lambda_j^{-i}(x_j) + \beta_{ij}(x_i, x_j) = \lambda_i^{-j}(x_i) + \beta_{ji}(x_j, x_i) = \frac{1}{2} \left( \theta_{ij}(x_i, x_j) + \lambda_i^{-j}(x_i) + \lambda_j^{-i}(x_j) \right) \tag{9}$$

which implies $\beta_{ij}(x_i, x_j) = \frac{1}{2} \left( \theta_{ij}(x_i, x_j) + \lambda_i^{-j}(x_i) - \lambda_j^{-i}(x_j) \right)$ and a similar expression for $\beta_{ji}$. The resulting $\lambda_{ij}(x_j) = \max_{x_i} \beta_{ij}(x_i, x_j)$ are then the ones in Prop. 2.

### Acknowledgments

The authors acknowledge support from the Defense Advanced Research Projects Agency (Transfer Learning program). Amir Globerson was also supported by the Rothschild Yad-Hanadiv fellowship.

## Footnotes

[1] We note that some authors also add a term $\sum_{i \in V} \theta_i(x_i)$ to $f(\boldsymbol{x}; \boldsymbol{\theta})$. However, these terms can be included in the pairwise functions $\theta_{ij}(x_i, x_j)$, so we ignore them for simplicity.

[2]As expected, NMPLP was faster than EMPLP so only NMPLP results are given.

[3]The edge weights for TRMP corresponded to a uniform distribution over all spanning trees.

[4]This is clearly a post-hoc measure since it can only be obtained after the algorithm has exceeded its maximum number of iterations. However, it is a reasonable algorithm-independent measure of convergence.

[5]The potential $\theta_i(x_i)$ may be folded into the pairwise potential to yield a model as in Eq. 1.

[6]Data available from http://jmlr.csail.mit.edu/papers/volume7/yanover06a/Rosetta_Design_Dataset.tgz

[7]Other solutions are possible but may not yield some of the properties of MPLP.

## References

[1] M. Bayati, D. Shah, and M. Sharma. Maximum weight matching via max-product belief propagation. *IEEE Trans. on Information Theory (to appear)*, 2007.

[2] D. P. Bertsekas, editor. *Nonlinear Programming*. Athena Scientific, Belmont, MA, 1995.

[3] A. Globerson and T. Jaakkola. Convergent propagation algorithms via oriented trees. In *UAI*. 2007.

[4] J.K. Johnson, D.M. Malioutov, and A.S. Willsky. Lagrangian relaxation for map estimation in graphical models. In *Allerton Conf. Communication, Control and Computing*, 2007.

[5] V. Kolmogorov. Convergent tree-reweighted message passing for energy minimization. *IEEE Transactions on Pattern Analysis and Machine Intelligence*, 28(10):1568–1583, 2006.

[6] V. Kolmogorov and M. Wainwright. On the optimality of tree-reweighted max-product message passing. In *21st Conference on Uncertainty in Artificial Intelligence (UAI)*. 2005.

[7] J. Pearl. *Probabilistic Reasoning in Intelligent Systems*. Morgan Kaufmann, 1988.

[8] B. Taskar, S. Lacoste-Julien, and M. Jordan. Structured prediction, dual extragradient and bregman projections. *Journal of Machine Learning Research*, pages 1627–1653, 2006.

[9] P.O. Vontobel and R. Koetter. Towards low-complexity linear-programming decoding. In *Proc. 4th Int. Symposium on Turbo Codes and Related Topics*, 2006.

[10] M. J. Wainwright, T. Jaakkola, and A. S. Willsky. Map estimation via agreement on trees: message-passing and linear programming. *IEEE Trans. on Information Theory*, 51(11):1120–1146, 2005.

[11] Y. Weiss, C. Yanover, and T. Meltzer. Map estimation, linear programming and belief propagation with convex free energies. In *UAI*. 2007.

[12] T. Werner. A linear programming approach to max-sum, a review. *IEEE Trans. on PAMI*, 2007.

[13] C. Yanover, T. Meltzer, and Y. Weiss. Linear programming relaxations and belief propagation – an empirical study. *Journal of Machine Learning Research*, 7:1887–1907, 2006.

[14] J.S. Yedidia, W.T. W.T. Freeman, and Y. Weiss. Constructing free-energy approximations and generalized belief propagation algorithms. *IEEE Trans. on Information Theory*, 51(7):2282–2312, 2005.

